# Informed Projections

**David Cohn**
Carnegie Mellon University
Pittsburgh, PA 15213
*cohn+@cs.cmu.edu*

## Abstract

Low rank approximation techniques are widespread in pattern recognition research — they include Latent Semantic Analysis (LSA), Probabilistic LSA, Principal Components Analysus (PCA), the Generative Aspect Model, and many forms of bibliometric analysis. All make use of a low-dimensional manifold onto which data are projected.

Such techniques are generally "unsupervised," which allows them to model data in the absence of labels or categories. With many practical problems, however, some prior knowledge is available in the form of context. In this paper, I describe a principled approach to incorporating such information, and demonstrate its application to PCA-based approximations of several data sets.

## 1 Introduction

Many practical problems involve modeling large, high-dimensional data sets to uncover similarities or latent structure. Linear low rank approximation techniques such as PCA [12], LSA [5], PLSA [6] and generative aspect models [1] are powerful tools for approaching these tasks. They identify (relatively) low-dimensional hyperplanes that best approximate the data according to a given noise model. In doing so, they exploit and expose regularities in the data: the hyperplanes represent a latent space whose dimensions are often observed to correspond to distinct latent categories in the data set. For example, an LSA-derived low-rank approximation to a corpus of news stories may have dimensions corresponding to "politics," "finance," "sports," etc. Documents with the same inferred sources (therefore "about" the same topic) generally lie close to each other in the latent space.

The broad applicability of these techniques comes from the fact that they are essentially "unsupervised" – a model is learned in the absence of labels indicating class or category memberships. There are, however, many situations in which some prior information is available; in these cases, we would like to have some way of using that information to improve our model.

Nigam et al. [10] studied the problem of learning to classify data into pre-existing categories in the presence of labeled and unlabeled examples. Their approach augmented a traditional supervised learning algorithm with distribution information made available from the unlabeled data. In contrast, this paper considers a method for augmenting a traditional unsupervised learning problem with the addition of equivalence classes.

Equivalence classes are a natural concept for many real-world problems. We frequently have some reason for believing that a set of observations are similar in some sense without wanting to or being able to say *why* they are similar. Note that the sets are not required to be comprehensive — we may only have known associations between a handful of observations. Further, the sets are not required to be disjoint; we may know that members of a set are similar, but there is no implication that members of two different sets are dissimilar.

In any case, the hope is that by indicating which observations are similar, we can bias our model focus on relevant features and to ignore differences that, while statistically significant, are not correlated with our idea of similarity in the problem at hand. This paper describes an algorithm validating the use of this approach.

## 1.1 Related work

There is too large a literature examining the combination of supervised and unsupervised learning to cover here; below I mention in passing some of the most relevant research.

In terms of conceptual similarity, multiple discriminant analysis (MDA) and oriented principal components analysis (OPCA) are techniques that attempt to maximize the fidelity of a linear low rank approximation while minimizing the variance of data belonging to designated equivalence classes [2]. The difference with the approach discussed here is that MDA and OPCA maximize a ratio of variances rather than a mixture; this is equivalent to making the assumption that the covariance matrices for each set are tied. Another related technique is multidimensional scaling (MDS) which, aside from sharing the ratio-based criterion, makes the added assumption that the precise degree of similarity (or dissimilarity) of two data points is to be enforced. In general, which set of assumptions is best depends on the problem at hand.

In terms of implementation, the present algorithm owes a great deal to the "shadow targets" algorithm for Neuroscale [8, 15], whose eponymous data points enforce equivalence classes on sets of (otherwise) unsupervised data. That algorithm trades fidelity of representation against fidelity of equivalence classes much in the same way as Equation 4, although it does so in the context of a Kohonen neural network instead of a linear mapping.

Another closely-related technique is CI-LSI [7], which uses latent semantic analysis for cross-language retrieval. The technique involves training on text documents from a parallel corpus for two or more languages (e.g. French and English), such that each document exists as both an English and French version. In CI-LSI, each document is merged with its twin, and the hyperplane is fit to the set of paired documents.

The goal of CI-LSI matches the goal of this paper, and the technique can in fact be seen as a special case of the informed projections discussed here. By using the "mean" of a pair of documents as a proxy for the documents themselves, we assert that the two come from a common source; fitting a model to a collection of such means finds a maximum likelihood solution subject to the constraint that both members of a pair comes from a common source.

## 2 Informed and uninformed projections

To introduce informed projections, I will first briefly review principal components analysis (PCA) and an algorithm for efficiently computing the principal components of a data set.

## 2.1 PCA and EMPCA

Given a finite data set $X \subset R^n$, where each column corresponds to one observation, PCA can be used to find a rank $m$ approximation $\hat{X}$ (where $m < n$) which minimizes the sum

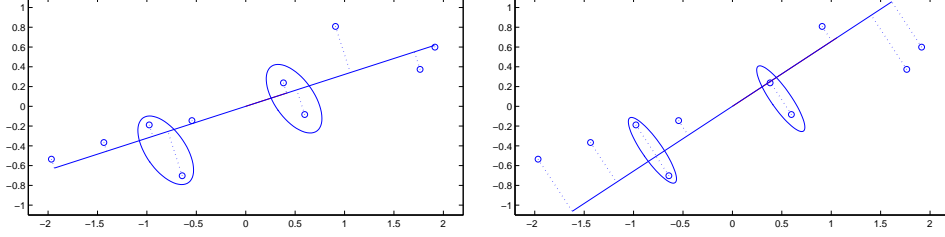

Figure 1: PCA maximizes the variance of the observations (on left), while an informed projection minimizes variance of projections from observations belonging to the same set.

squared distortion with respect to $X$. It does this by identifying the $m$ orthogonal directions in which $X$ exhibits the greatest variance, corresponding to the $m$ largest eigenvectors $C = [C_1, \ldots, C_m]$. $X$ can then be projected onto the hyperplane defined by $C$ as

$$\hat{X} = C(C^T C)^{-1} C^T X. \tag{1}$$

Although not strictly a generative model, PCA offers a probabilistic interpretation: $C$ represents a maximum likelihood model of the data under the assumption that $X$ consists of (Gaussian) noise-corrupted observations taken from linear combinations of $m$ sources in an $n$-dimensional space. The values for $\hat{X}$ then represent maximum likelihood estimates of the mixtures responsible for the corresponding values in $X$.

Roweis [13] described an efficient iterative technique for identifying $C$ using an EM procedure. Beginning with an arbitrary guess for $C$, the latent representation of $X$ is computed

$$Y = (C^T C)^{-1} C^T X \tag{2}$$

after which $C$ is updated to maximize the estimated likelihoods

$$C = XY^T (YY^T)^{-1}. \tag{3}$$

Equations 2 and 3 are iterated until convergence (typically less than 10 iterations), at which time the sum squared error of $\hat{X}$'s approximation to $X$ will have been minimized.

## 2.2 Informed projections

PCA only penalizes according to squared distance of an observation $x_i$ from its projection $\hat{x}_i$. Given a Gaussian noise model, $\hat{x}_i$ is the maximum likelihood estimate of $x_i$'s "source," which is the only constraint with which PCA is concerned.

If we believe that a set of observations $S_i = \{x_1, x_2, \ldots, x_n\}$ have a common cause, then they should share a common source. For a hyperplane defined by eigenvectors $C$, the maximum likelihood source is the mean of $S_i$'s projections onto $C$, denoted $\overline{S}_i$. As such, the likelihood should be penalized not only on the basis of the variance of observations around their projections $\left( \sum_j ||x_j - \hat{x}_j||^2 \right)$, but also the variance of the projections around their set means $\left( \sum_i \sum_{x_j \in S_i} ||\hat{x}_j - \overline{S}_i||^2 \right)$.

These two penalty terms may be at odds with each other, so we must introduce a hyperparameter $\beta$ representing how much weight to place on accurately reproducing the original observations and how much to place on preserving the integrity of the known sets:

$$E_\beta = (1 - \beta) \sum_j ||x_j - \hat{x}_j||^2 + \beta \sum_i \sum_{x_j \in S_i} ||\hat{x}_j - \overline{S}_i||^2. \tag{4}$$

When $\beta = 0.5$, Equation 4 is equivalent to minimizing $\sum_i \sum_{x_j \in S_i} ||x_j - \overline{S}_i||^2$ under the assumption that all otherwise unaffiliated $x_i$ are members of their own singleton sets. This is just the squared distance from each observation to its projected cluster mean, which appears to be the criterion CI-LSI minimizes by averaging documents.

## 2.3 Finding an informed projection

The error criterion in 4 may be efficiently optimized with an expectation-maximization (EM) procedure based on Roweis' EMPCA [13], alternately computing estimated sources $\hat{x}$ and maximizing the likelihoods of the observed data given those sources.

The likelihood of a set is maximized by minimizing the variance of projections from members of a set around their mean. This is at odds with the efforts of PCA to maximize likelihood by maximizing the variance of projections from the data set at large. We can make these forces work together by adding a "complement set" $\tilde{S}_i$ for each set $S_i$ such that the variance of $S_i$'s projections is minimized by maximizing the variance of $\tilde{S}_i$'s projections.

The complement set may be determined analytically, but can also be computed efficiently as an extra step between the "E" and "M" steps of the EM iteration. Given an observation $x_j \in S_i$, the complement for $x_j$ may be computed in terms of its projection $\hat{x}_j$ onto the hyperplane and $\overline{S}_i$, the mean of the set.

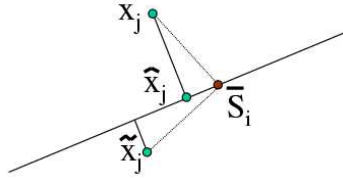

Figure 2: Location of a point's complement $\tilde{x}_j$ with respect to its mean set projection $\overline{S}_i$ and the current hyperplane.

In order to "pull" the current hyperplane in the direction that will minimize $x_j$'s distance from the set mean, $\tilde{x}_j$ must be positioned at a distance of $||x_j - \hat{x}_j||$ from the hyperplane such that its projection lies along line from $\overline{S}_i$ to $\hat{x}_j$ at a distance from $\overline{S}_i$ equal to $||x_j - \hat{x}_j||$. With some geometric manipulation (Figure 2), it can be shown that

$$\tilde{x}_j = \overline{S}_i + (\hat{x}_j - \overline{S}_i)\frac{||\hat{x}_j - x||}{||\hat{x}_j - \overline{S}_i||} + (\hat{x}_j - x)\frac{||\hat{x}_j - \overline{S}_i||}{||\hat{x}_j - x||}.$$

For efficiency, it is worth noting that by subtracting each set's mean from its constituent observations, all sets may be combined into a single zero-mean "superset" $\tilde{S}$ from which complements are computed.

Once the complement set has been computed, it can be appended to the original observations a to create a joint data set, denoted $X^+ = [X|\tilde{S}]$, and the "M" step of the EM procedure is continued as before:[1]

$$Y = (C^T C)^{-1} C^T X^+, \quad C = X^+ Y^T (YY^T)^{-1}. \tag{5}$$

Applying $\beta$ to the optimization is straightforward – if we preprocess the data by subtracting the mean of the observations (as is standard for PCA), the effect of each observation is to

apply a "torque" to the current hyperplane around the origin. By multiplying all coordinates of an observation by the same scalar, we scale the torque applied by the same amount. As such, we can trade off the weight attached to enforcing the sets against the weight attached to reconstructing the original data by multiplying $\tilde{S}$ and $X$ by $\beta$ and $1-\beta$ respectively:

$$X_\beta^+ = [(1-\beta)X | \beta \cdot \tilde{S}]$$

## 3   Experiments

I examined the effect of "informing" projections on three data sets from two domains. The first two were text data sets taken from the WebKB project and the "20 newsgroups" data set. The third data set consisted of acoustic features from recorded music. Finally, I examine the effect of adding set information to the joint probabilistic model described by Cohn and Hofmann [3].

### 3.1   WebKB data

The first set of experiments began with a subset of the WebKB data set [4]. Using Rainbow [9], I tokenized 1000 randomly-selected documents, stripping out HTML and digits, and kept the 1000 terms with highest class-dependent information gain (the reduced vocabulary greatly decreased processing times). The result was 1000 documents with 1000 features, where feature $f_{i,j}$ represented the frequency with which term $j$ occurred in document $x_i$. Sets were constructed from the categories provided with each document.

The experiments varied both the fraction of the training data for which set associations were provided (0-1) and the weight given to preserving those sets (also 0-1). For each combination, I ran 40 trials, each using a randomized split of 200 training documents and 100 test documents. Accuracy was evaluated based on leave-one-out nearest neighbor classification over the test set.[2]

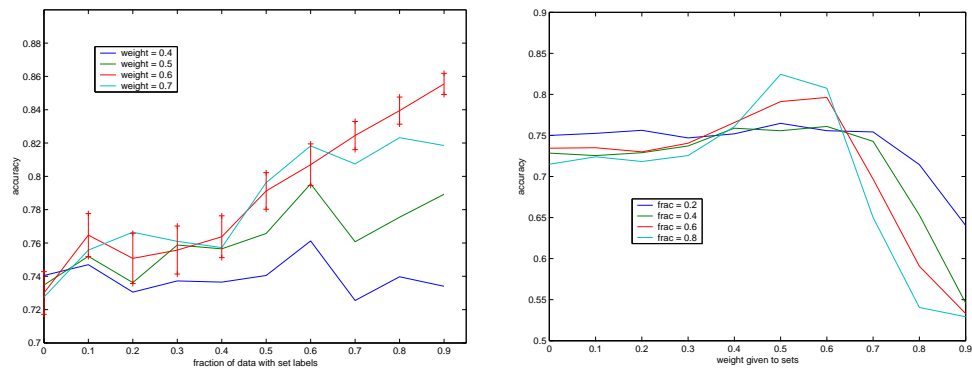

Figure 3: Nearest neighbor classification of WebKB data, where a 5D PCA of document terms has been informed by web page category-determined sets (40 independent train/test splits). The fraction of observations that have been given set assignments is varied from 0 to 1 (left plot), as is $\beta$, the weight attached to preserving set associations (right plot).

Figure 3 summarizes the results of these experiments. As expected, the more documents that had set associations, the greater the improvement in classification accuracy, but this

improvement was only evident for $0.3 \leq \beta \leq 0.7$; below 0.3, the sets were not given enough weight to make a difference, while above 0.7 there is a rapid deterioration in accuracy.

## 3.2   20 Newsgroups

The second set of experiments also used a standard text classification corpus, but with an unrestricted vocabulary. Beginning with the documents of the 20 newsgroups data set, I again preprocessed the documents as above with Rainbow, but this time kept the entire vocabulary (27214 unique terms), instead of preselecting maximally informative terms.

Because of the additional running time required to handle the complete vocabularies, the experiments used all set labels and only varied the weighting. Thirty independent training and test sets of 100 documents each were run for $0 \leq \beta \leq 1$, and as before, accuracy was eveluted in terms of leave-one-out classification error on the test set.

Figure 4 summarizes the results of these experiments. The characteristic learning curve is very similar to that for the WebKB data — an interme-

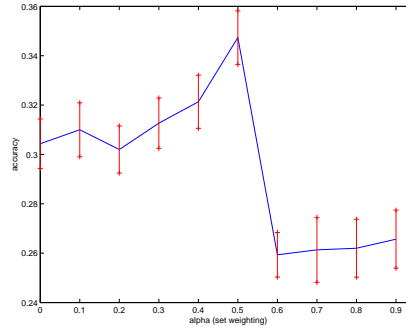

Figure 4: Five categories from 20 newsgroups data set, where a 5D PCA of document terms has been informed by source category (30 train/test splits, for $0 < \beta < 1$).

diate set weighting yields significantly better performance than the purely supervised or unsupervised cases. There is, however, one notable distinction: in these experiments, there is much less variation in accuracy for large values of $\beta$ — it almost appears that there are three stable regions of performance.

## 3.3   Album recognition from acoustic features

The third test used a proprietary data set of acoustic properties of recorded music. The data set contained 11252 recorded music tracks from 939 albums. Each observation consisted of 85 highly-processed acoustic features extracted automatically via digital signal processing.

The goal of this experiment was to determine whether informing a projected model could improve the accuracy with which it could identify tracks from the same album. Recalling Platt's playlist selection problem [11], this can serve as a proxy for estimating how well the model can predict whether two tracks "belong together" by the subjective measure of the artist who created the album.

For these experiments, I selected the first 8439 tracks (3/4 of the data) for training, assigning each track to be a member of the set defined by the album it came from. Many tracks appeared on multiple albums ("Best of..." and soundtrack collections). The remaining 2813 tracks were used as test data.

The 85 dimensional features were projected down into a 10 dimensional space, informing the projection with sets defined by tracks from the same album. The relatively low dimension of the problem permitted also running OPCA on the data set for comparison. As above, I measured the frequency with which each test track had another track from the same album as its nearest neighbor when projected down into this same space.

While the improvements in performance are not as striking as those from the previous experiments, they are nonetheless significant (Table 1). One reason for the meager improvement may be that the features from which the projections were computed had already been

| weight | β = 0.0 | β = 0.5 | β = 1.0 | OPCA |
|---|---|---|---|---|
| accuracy | 0.1070 | 0.1241 | 0.0551 | 0.1340 |
| ratio | 0.3859 | 0.3223 | 0.3414 | 0.3144 |

Table 1: Album recognition results using 2813 test tracks from 316 albums. For each weighting β, "accuracy" is the fraction of times which the closest track to a test track came from the same album; "ratio" indicates the average ratio of intra-album distances to inter-album distances in the test set. In all cases, informing the projection with a weight of β = 0.5 increases the accuracy and decreases the ratio of the model.

manually optimized for classification accuracy. Interestingly, OPCA slightly outperforms the informed projection for both criteria on this problem.

### 3.4 Content, context and connections

Prior work [3] discussed building joint probabilistic models of a document base, using both the content of the documents and the connections (citations or hyperlinks) between them. A document base frequently contains context as well, in the form of documents from the same source or by the same author. Informed projection provides a way for us to inject this third form of information and further improve our models.

Figure 5 summarizes the results of using set information to "inform" the joint content+link models discussed in the previous paper. That work used a multinomial model for its approximation, so we can not use the equations defined in Section 2.3. Instead, we can make use of the observation of Section 1.1 to approximate the informing process by merging documents from the same set. Figure 5 illustrates that this process complements the earlier content+connections approach, providing a joint model of document content, context and connections.

| accuracy (std err) | uninformed | informed |
|---|---|---|
| content | 0.19 (0.017) | 0.33 (0.039) |
| links | 0.11 (0.013) | 0.23 (0.098) |
| both | 0.21 (0.023) | 0.33 (0.057) |

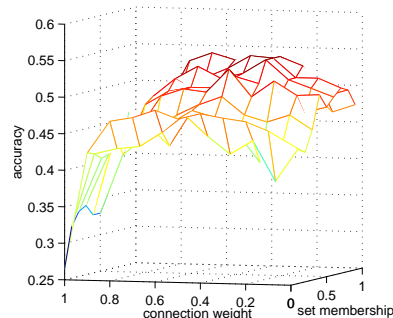

Figure 5: (left) Classification accuracy of informed vs. uninformed models of separate and joint models of document content and connections, using the WebKB dataset. (right) Effect of adding more document context in the form of set membership information on the Cora data set. See Cohn and Hofmann [3] for details.

## 4 Discussion and future work

The experiments so far indicate that adding set information to a low rank approximation does improve the quality of a model, but only to the extent that the information is used in conjunction with the unsupervised information already present in the data set. The improvement in performance is evident for content models (such as LSA), connection models, and joint models of content and connections.

## 4.1 Future work

Beyond experiments that to clarify the effect of $\beta$ on model fitness, there are many obvious directions for future work. The first is further exploration on the relationship between informed PCA and and the variants of MDA discussed in Section 1.1. While the differences are mathematically straightforward, the effect of sum-vs.-ratio criteria bears further study.

A second broad area for future work is the application of the techniques described here to richer low rank approximation models. While this paper considered the effect of informing PCA, it would be fruitful to examine both the process and effect of informing multinomial-based models [3, 6], fully-generative models [1] and local linear embeddings [14].

A third area for exploration is the study of potential applications for this approach, which include improved relevance modeling, directed web crawling, and personalized search and recommendation across a wide variety of media.

## Footnotes

[1]Since $\tilde{S}_i$ depends on the projections, and therefore the position of the hyperplane, it must be recomputed with each iteration.

[2]Obviously, simple nearest neighbor is far from the most effective classification technique for this domain. But the point of the experiment is to evaluate to what degree informing a projection preserves or improves topic locality, which nearest neighbor classifiers are well-suited to measure.

## References

[1] D. Blei, A. Ng, and M. I. Jordan. Latent dirichlet allocation. In *Advances in Neural Information Processing Systems 14*, 2002.

[2] C.J.C. Burges, J.C. Platt, and S. Jana. Extracting noise-robust features from audio data. In *Proceedings of ICASSP*, 2002.

[3] D. Cohn and T. Hofmann. The missing link - a probabilistic model of document content and hypertext connectivity. In T. Leen et al., editor, *Advances in Neural Information Processing Systems 13*, 2001.

[4] M. Craven, D. DiPasquo, D. Freitag, A. McCallum, T. Mitchell, K. Nigam, and S. Slattery. Learning to extract symbolic knowledge from the world wide web. In *Proceedings of the 15th National Conference on Artificial Intelligence (AAAI-98)*, 1998.

[5] S. Dumais, G. Furnas, T. Landauer, S. Deerwester, and R. Harshman. Using latent semantic analysis to improve access to textual information. In *Proceedings of the Conference on Human Factors in Computing Systems CHI'88*, 1988.

[6] T. Hofmann. Probabilistic latent semantic analysis. In *Proc. of Uncertainty in Artificial Intelligence, UAI'99*, Stockholm, 1999.

[7] M. Littman, S. Dumais, and T. Landauer. Automatic cross-language information retrieval using latent semantic indexing. In G. Grefenstette, editor, *Cross Language Information Retrieval*. Kluwer, 1998.

[8] D. Lowe and M. E. Tipping. Feed-forward neural networks and topographic mappings for exploratory data analysis. *Neural Computing and Applications*, 4:83–95, 1996.

[9] A. K. McCallum. Bow: A toolkit for statistical language modeling, text retrieval, classification and clustering. http://www.cs.cmu.edu/ mccallum/bow, 1996.

[10] K. Nigam, A. K. McCallum, S. Thrun, and T. M. Mitchell. Learning to classify text from labeled and unlabeled documents. In *Proceedings of AAAI-98*, pages 792–799, Madison, US, 1998. AAAI Press, Menlo Park, US.

[11] J. Platt, C. Burges, S. Swenson, C. Weare, and A. Zheng. Learning a gaussian process prior for automatically generating music playlists. In T. G. Dietterich, S. Becker, and Z. Ghahramani, editors, *Advances in Neural Information Processing Systems 14*. MIT Press, 2002.

[12] B. D. Ripley. *Pattern Recognition and Neural Networks*. Cambridge: University Press, 1996.

[13] S. Roweis. EM algorithms for PCA and SPCA. In M. I. Jordan, M. J. Kearns, and S. A. Solla, editors, *Advances in Neural Information Processing Systems*, volume 10. MIT Press, 1998.

[14] S. Roweis and L. Saul. Nonlinear dimensionality reduction by locally linear embedding. *Science*, 290(5500):2323–2326, Dec 2000.

[15] M. E. Tipping and D. Lowe. Shadow targets: A novel algorithm for topographic projections by radial basis functions. *Neurocomputing*, 19(1):211–222, 1998.
